# Cocktail Party Processing via Structured Prediction

**Yuxuan Wang**[1] **, DeLiang Wang**[1,2]
[1]Department of Computer Science and Engineering
[2]Center for Cognitive Science
The Ohio State University
Columbus, OH 43210
{wangyuxu,dwang}@cse.ohio-state.edu

## Abstract

While human listeners excel at selectively attending to a conversation in a cocktail party, machine performance is still far inferior by comparison. We show that the cocktail party problem, or the speech separation problem, can be effectively approached via structured prediction. To account for temporal dynamics in speech, we employ conditional random fields (CRFs) to classify speech dominance within each time-frequency unit for a sound mixture. To capture complex, nonlinear relationship between input and output, both state and transition feature functions in CRFs are learned by deep neural networks. The formulation of the problem as classification allows us to directly optimize a measure that is well correlated with human speech intelligibility. The proposed system substantially outperforms existing ones in a variety of noises.

## 1 Introduction

The cocktail party problem, or the speech separation problem, is one of the central problems in speech processing. A particularly difficult scenario is monaural speech separation, in which mixtures are recorded by a single microphone and the task is to separate the target speech from its interference. This is a severely underdetermined figure-ground separation problem, and has been studied for decades with limited success.

Researchers have attempted to solve the monaural speech separation problem from various angles. In signal processing, speech enhancement (e.g., [1, 2]) has been extensively studied, and assumptions regarding the statistical properties of noise are crucial to its success. Model-based methods (e.g., [3]) work well in constrained environments, and source models need to be trained in advance. Computational auditory scene analysis (CASA) [4] is inspired by how human auditory system functions [5]. CASA has the potential to deal with general acoustic environments but existing systems have limited performance, particularly in dealing with unvoiced speech.

Recent studies suggest a new formulation to the cocktail party problem, where the focus is to classify whether a time-frequency (T-F) unit is dominated by the target speech [6]. Motivated by this viewpoint, we propose to approach the monaural speech separation problem via structured prediction. The use of structured predictors, as opposed to binary classifiers, is motivated by temporal dynamics in speech signal. Our study makes the following contributions: (1) we demonstrate that modeling temporal dynamics via structured prediction can significantly improve separation; (2) to capture nonlinearity, we propose a new structured prediction model that makes use of the discriminative feature learning power of deep neural networks; and (3) instead of classification accuracy, we show how to directly optimize a measure that is well correlated with human speech intelligibility.

## 2  Separation as binary classification

We aim to estimate a time-frequency matrix called the ideal binary mask (IBM). The IBM is a binary matrix constructed from premixed target and interference, where 1 indicates that the target energy exceeds the interference energy by a local signal-to-noise (SNR) criterion (LC) in the corresponding T-F unit, and 0 otherwise. The IBM is defined as:

$$IBM(t, f) = \begin{cases} 1, & \text{if } SNR(t, f) > LC \\ 0, & \text{otherwise,} \end{cases}$$

where $SNR(t, f)$ denotes the local SNR (in decibels) within the T-F unit at time $t$ and frequency $f$. We adopt the common choice of $LC = 0$ in this paper [7]. Despite its simplicity, adopting the IBM as a computational objective offers several advantages. First, the IBM is directly based on the auditory masking phenomenon whereby a stronger sound tends to mask a weaker one within a critical band. Second, unlike other objectives such as maximizing SNR, it is well established that large human speech intelligibility improvements result from IBM processing, even for very low SNR mixtures [7–9]. Improving human speech intelligibility is considered as a gold standard for speech separation. Third, IBM estimation naturally leads to classification, which opens the cocktail party problem to a plethora of machine learning techniques.

We propose to formulate IBM estimation as binary classification as follows, which is a form of supervised learning. A sound mixture with the 16 kHz sampling rate is passed through a 64-channel gammatone filterbank spanning from 50 Hz to 8000 Hz on the equivalent rectangular bandwidth rate scale. The output from each filter channel is divided into 20-ms frames with 10-ms frame shift, producing a cochleagram [4]. Due to different spectral properties of speech, a subband classifier is trained for each filter channel independently, with the IBM providing training labels. Acoustic features for each subband classifier are extracted from T-F units in the cochleagram. The target speech is separated by binary weighting of the cochleagram using the estimated IBM [4].

Several recent studies have attempted to directly estimate the IBM via classification. By employing Gaussian mixture models (GMMs) as classifiers and amplitude modulation spectrograms (AMS) as features, Kim et al. [10] show that estimated masks can improve human speech intelligibility in noise. Han and Wang [11] have improved Kim et al.'s system by employing support vector machines (SVMs) as classifiers. Wang et al. [12] propose a set of complementary acoustic features that shows further improvements over previous systems. The complementary feature is a concatenation of AMS, relative spectral transform and perceptual linear prediction (RASTA-PLP), mel-frequency cepstral coefficients (MFCC), and pitch-based features.

Because the ratio of 1's to 0's in the IBM is often skewed, simply using classification accuracy as the evaluation criterion may not be appropriate. Speech intelligibility studies [9, 10] have evaluated the influence of the hit (HIT) and false-alarm (FA) rate on intelligibility scores. The difference, the HIT−FA rate, is found to be well correlated to human speech intelligibility in noise [10]. The HIT rate is the percent of correctly classified target-dominant T-F units (1's) in the IBM, and the FA rate is the percent of wrongly classified interference-dominant T-F units (0's). Therefore, it is desirable to design a separation algorithm that maximizes HIT−FA of the output mask.

## 3  Proposed system

Dictated by speech production mechanisms, the IBM contains highly structured, rather than, random patterns. Previous systems do not explicitly model such structure. As a result, temporal dynamics, which is a fundamental characteristic of speech, is largely ignored in previous work. Separation systems accounting for temporal dynamics exist. For example, Mysore et al. [13] incorporate temporal dynamics using HMMs. Hershey et al. [14] consider different levels of dynamic constraints. However, these works do not treat separation as classification. Contrary to standard binary classifiers, structured prediction models are able to model correlations in the output. In this paper, we treat unit classification at each filter channel as a sequence labeling problem and employ linear-chain conditional random fields (CRFs) [15] as subband classifiers.

## 3.1 Conditional random fields

Different from HMM, a CRF is a discriminative model and does not need independence assumptions of features, making it more suitable to our task. A CRF models the posterior probability $P(\mathbf{y}|\mathbf{x})$ as follows. Denoting $\mathbf{y}$ as a label sequence and $\mathbf{x}$ as an input sequence,

$$P(\mathbf{y}|\mathbf{x}) = \frac{\exp\left(\sum_t \mathbf{w}^T \mathbf{f}(\mathbf{y}, \mathbf{x}, t)\right)}{Z(\mathbf{x})}. \tag{1}$$

Here $t$ indexes time frames, $\mathbf{w}$ is the parameters to learn, and $Z(\mathbf{x}) = \sum_{\mathbf{y}'} \exp\left(\sum_t \mathbf{w}^T \mathbf{f}(\mathbf{y}', \mathbf{x}, t)\right)$ is the partition function. $\mathbf{f}$ is a vector-valued feature function associated with each local site (T-F unit in our task), and often categorized into state feature functions $\mathbf{s}(y_t, \mathbf{x}, t)$ and transition feature functions $\mathbf{t}(y_{t-1}, y_t, \mathbf{x}, t)$. State feature functions define the local discriminant functions for each T-F unit and transition feature functions capture the interaction between neighboring labels. We assume a linear-chain setting and the first-order Markovian property, i.e., only interactions between two neighboring units in time are modeled. In our task, we can simply use acoustic feature vectors in each T-F unit as state feature functions and their concatenations as transition feature functions:

$$\mathbf{s}(y_t, \mathbf{x}, t) = [\delta_{(y_t=0)}\mathbf{x}_t, \delta_{(y_t=1)}\mathbf{x}_t]^T, \tag{2}$$

$$\mathbf{t}(y_{t-1}, y_t, \mathbf{x}, t) = [\delta_{(y_{t-1}=y_t)}\mathbf{z}_t, \delta_{(y_{t-1}\neq y_t)}\mathbf{z}_t]^T, \tag{3}$$

where $\delta$ is the indicator function and $\mathbf{z}_t = [\mathbf{x}_{t-1}, \mathbf{x}_t]^T$. Equation (3) essentially encodes temporal continuity in the IBM. To simplify notations, all feature functions are written as $\mathbf{f}(y_{t-1}, y_t, \mathbf{x}, t)$ in the remainder of the paper.

Training is for estimating $\mathbf{w}$, and is usually done by maximizing the conditional log-likelihood on a training set $T = \left\{\left(\mathbf{x}^{(m)}, \mathbf{y}^{(m)}\right)\right\}$, i.e., we seek $\mathbf{w}$ by

$$\max_{\mathbf{w}} \sum_m \log p(\mathbf{y}^{(m)}|\mathbf{x}^{(m)}, \mathbf{w}) + \mathcal{R}(\mathbf{w}), \tag{4}$$

where $m$ is the index of a training sample, and $\mathcal{R}(\mathbf{w})$ is a regularizer of $\mathbf{w}$ (we use $\ell_2$ in this paper). For gradient ascent, a popular choice is the limited-memory BFGS (L-BFGS) algorithm [16].

## 3.2 Nonlinear expansion using deep neural networks

A CRF is a log-linear model, which has only linear modeling power. As acoustic features are generally not linearly separable, the direct use of CRFs unlikely produces good results. In the following, we propose a method to transform the standard CRF into a nonlinear sequence classifier.

We employ pretrained deep neural networks (DNNs) to capture nonlinearity between input and output. DNNs have received widespread attention since Hinton et al.'s paper [17]. DNNs can be viewed as hierarchical feature detectors that learn increasingly complex feature mappings as the number of hidden layers increases. To deal with problems such as vanishing gradients, Hinton et al. suggest to first pretrain a DNN using a stack of restricted Boltzmann machines (RBMs) in a unsupervised and layerwise fashion. The resulting network weights are then supervisedly finetuned by backpropagation.

We first train DNN in the standard way to classify speech dominance in each T-F unit. After pretraining and supervised finetuning, we then take the last hidden layer representations from the DNN as learned features to train the CRF. In a discriminatively trained DNN, the weights from the last hidden layer to the output layer would define a linear classifier, hence the last hidden layer representations are more amenable to linear classification. In other words, we replace $\mathbf{x}$ by $\mathbf{h}$ in equations (1)-(4), where $\mathbf{h}$ represents the learned hidden features. This way CRFs would greatly benefit from the nonlinear modeling power of deep architectures.

To better encode local contextual information, we could use a window (across both time and frequency) of learned features to label the current T-F unit. A more parsimonious way is to use a window of posteriors estimated by DNNs as features to train the CRF, which can dramatically reduce the dimensionality. We note in passing that the correlations across both time and frequency can also be encoded at the model level, e.g., by using grid-structured CRFs. However the decoding algorithm may substantially increase the computational complexity of the system.

We want to point out that an important advantage of using neural networks for feature learning is its efficiency in the test phase; once trained, the nonlinear feature extraction of DNN is extremely fast (only involves forward pass). This is, however, not always true for other methods. For example, sparse coding may need to solve a new optimization problem to get the features. Test phase efficiency is crucial for real-time implementation of a speech separation system.

There is related work on developing nonlinear sequence classifiers in the machine learning community. For example, van der Maaten et al. [18] and Morency et al. [19] consider incorporating hidden variables into the training and inference in CRF. Peng et al. [20] investigate a combination of neural networks and CRFs. Other related studies include [21] and [22]. The proposed model differs from the previous methods in that (1) discriminatively trained deep architecture is used, and/or (2) a CRF instead of a Viterbi decoder is used on top of a neural network for sequence labeling, and/or (3) nonlinear features are also used in modeling transitions. In addition, the use of a contextual window and the change of the objective function discussed in the next subsection is specifically tailored to the speech separation problem.

## 3.3 Maximizing HIT−FA rate

As argued before, it is desirable to train a classifier to maximize the HIT−FA rate of the estimated mask. In this subsection, we show how to change the objective function and efficiently calculate the gradients in CRF. Since subband classifiers are used, we aim to maximize the channelwise HIT−FA.

Denote the output label as $u_t \in \{0, 1\}$ and the true label as $y_t \in \{0, 1\}$. The per utterance HIT−FA rate can be expressed as $\sum_t u_t y_t / \sum_t y_t - \sum_t u_t(1 - y_t) / \sum_t (1 - y_t)$, where the first term is the HIT rate and the second the FA rate. To make the objective function differentiable, we replace $u_t$ by the marginal probability $p(y_t = 1|\mathbf{x})$, hence we seek $\mathbf{w}$ by maximizing the HIT−FA on a training set:

$$\max_{\mathbf{w}} \left( \frac{\sum_m \sum_t p(y_t^{(m)} = 1|\mathbf{x}^{(m)}, \mathbf{w}) y_t^{(m)}}{\sum_m \sum_t y_t^{(m)}} - \frac{\sum_m \sum_t p(y_t^{(m)} = 1|\mathbf{x}^{(m)}, \mathbf{w})(1 - y_t^{(m)})}{\sum_m \sum_t (1 - y_t^{(m)})} \right). \quad (5)$$

Clearly, computing the gradient of (5) boils down to computing the gradient of the marginal. A speech utterance (sentence) typically spans several hundreds of time frames, therefore numerical stability is critically important in our task. As can be seen later, computing the gradient of the marginal requires the gradient of forward/backward scores. We adopt Rabiner's scaling trick [23] used in HMM to normalize the forward/backward score at each time point. Specifically, define $\alpha(t, u)$ and $\beta(t, u)$ as the forward and backward score of label $u$ at time $t$, respectively. We normalize the forward score such that $\sum_u \alpha(t, u) = 1$, and use the resulting scaling to normalize the backward score. Defining potential function $\phi_t(v, u) = \exp\left(\mathbf{w}^T \mathbf{f}(v, u, \mathbf{x}, t)\right)$, the recurrence of the normalized forward/backward score is written as,

$$\alpha(t, u) = \sum_v \alpha(t - 1, v)\phi_t(v, u)/s(t), \quad (6)$$

$$\beta(t, u) = \sum_v \beta(t + 1, v)\phi_t(u, v)/s(t + 1), \quad (7)$$

where $s(t) = \sum_u \sum_v \alpha(t - 1, v)\phi_t(v, u)$. It is easy to show that $Z(\mathbf{x}) = \prod_t s(t)$, and now the marginal has a simpler form of $p(y_t|\mathbf{x}, \mathbf{w}) = \alpha(t, y_t)\beta(t, y_t)$. Therefore, the gradient of the marginal is,

$$\frac{\partial p(y_t|\mathbf{x}, \mathbf{w})}{\partial \mathbf{w}} = G_\alpha(t, y_t)\beta(t, y_t) + \alpha(t, y_t)G_\beta(t, y_t), \quad (8)$$

where $G_\alpha$ and $G_\beta$ are the gradients of the normalized forward and backward score, respectively. Due to score normalization, $G_\alpha$ and $G_\beta$ will very unlikely overflow. We now show that $G_\alpha$ can be calculated recursively. Let $q(t, u) = \sum_v \alpha(t - 1, v)\phi_t(v, u)$, we have

$$G_\alpha(t, u) = \frac{\partial \alpha(t, u)}{\partial \mathbf{w}} = \frac{\frac{\partial q(t,u)}{\partial \mathbf{w}} \sum_v q(t, v) - \sum_v \frac{\partial q(t,v)}{\partial \mathbf{w}} q(t, u)}{\left(\sum_v q(t, v)\right)^2}, \quad (9)$$

and,

$$\frac{\partial q(t, u)}{\partial \mathbf{w}} = \sum_v G_\alpha(t - 1, v)\phi_t(v, u) + \sum_v \alpha(t - 1, v)\phi_t(v, u)\mathbf{f}(v, u, \mathbf{x}, t). \quad (10)$$

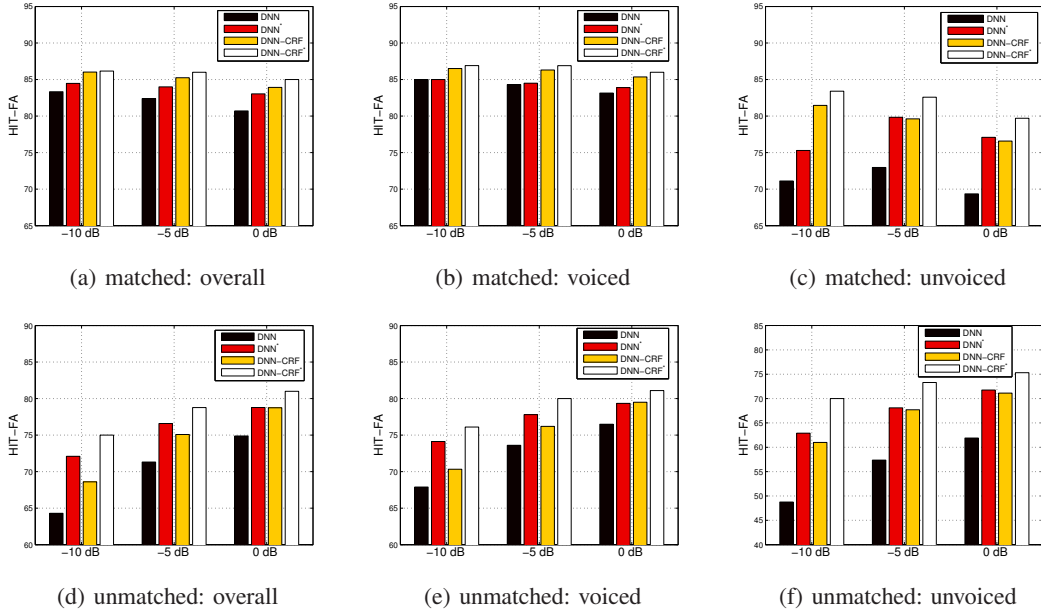

(a) matched: overall      (b) matched: voiced      (c) matched: unvoiced

(d) unmatched: overall      (e) unmatched: voiced      (f) unmatched: unvoiced

Figure 1: HIT−FA results. (a)-(c): matched-noise test condition; (d)-(f): unmatched-noise test condition.

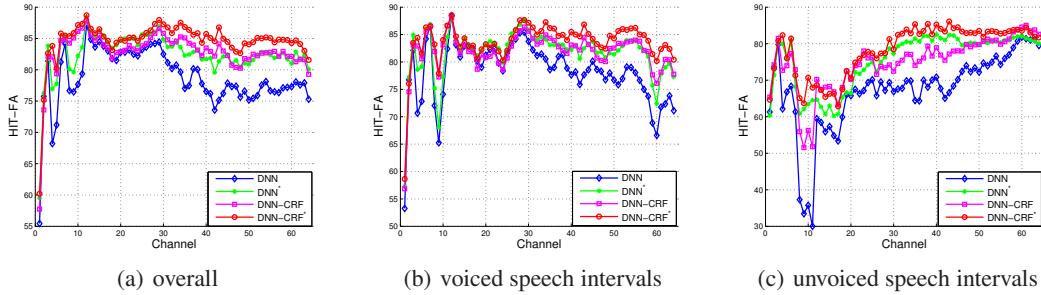

(a) overall      (b) voiced speech intervals      (c) unvoiced speech intervals

Figure 2: Channelwise HIT−FA comparisons on the 0 dB test mixtures.

The derivation of $G_\beta$ is similar, thus omitted. The time complexity of calculating $G_\alpha$ and $G_\beta$ is $\mathcal{O}(L|S|^2)$, where $L$ and $|S|$ are the utterance length and the size of the label set, respectively. This is the same as the forward-backward recursion.

The objective function in (5) is not concave. Since high accuracy correlates with high HIT−FA, a safe practice is to use a solution from (4) as a warm start for the subsequent optimization of (5). For feature learning, DNN is also trained using (5) in the final system. The gradient calculation is much simpler due to the absence of transition features. We found that L-BFGS performs well and shows fast and stable convergence for both feature learning and CRF training.

## 4 Experimental results

### 4.1 Experimental setup

Our training and test sets are primarily created from the IEEE corpus [24] recorded by a single female speaker. This enables us to directly compare with previous intelligibility studies [10], where the same speaker is used in training and testing. The training set is created by mixing 50 utterances with 12 noises at 0 dB. To create the test set, we choose 20 unseen utterances from the same speaker. First, the 20 utterances are mixed with the previous 12 noises to create a matched-noise test

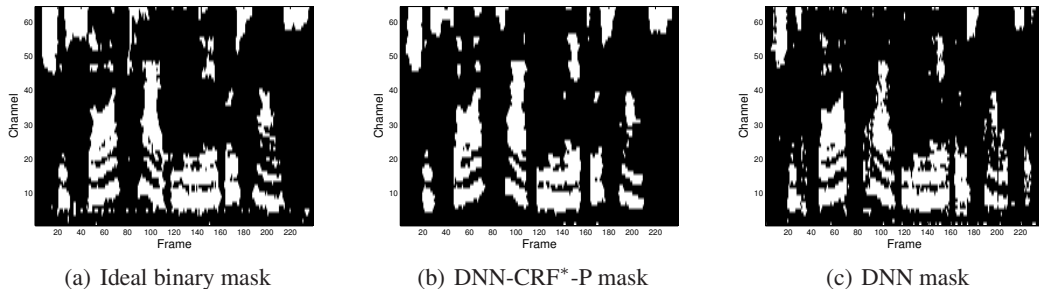

| (a) Ideal binary mask | (b) DNN-CRF*-P mask | (c) DNN mask |

Figure 3: Masks for a test utterance mixed with an unseen crowd noise at 0 dB. White represents 1's and black represents 0's.

condition, then 5 unseen noises to create a unmatched-noise test condition. The test noises[1] cover a variety of daily noises and most of them are highly non-stationary. In each frequency channel, there are roughly 150,000 and 82,000 T-F units in the training and test set, respectively. Speaker-independent experiments are presented in Section 4.4.

The proposed system is called DNN-CRF or DNN-CRF* if it is trained to maximize HIT−FA. We use suffix R and P to distinguish training features for CRF, where R stands for learned features without a context window (features are learned from the complementary acoustic feature set mentioned in Section 2) and P stands for a window of posterior features. We use a two hidden layer DNN as it provides a good trade-off between performance and complexity, and use a context window spanning 5 time frames and 17 frequency channels to construct the posterior feature vector. We use the cross-entropy objective function for training the standard DNN in comparisons.

## 4.2 Experiment 1: HIT−FA maximization

In this subsection, we show the effect of directly maximizing the HIT−FA rate. To evaluate the contribution from the change of the objective alone, we use ideal pitch in the following experiments to neutralize pitch estimation errors. The models are trained on 0 dB mixtures. In addition to 0 dB, we also test the trained models on -10 and -5 dB mixtures. Such a test setting not only allows us to measure the system's generalization to different SNR conditions, but also to show the effects of HIT−FA maximization on estimating sparse IBMs. We compare DNN-CRF*-R with DNN, DNN* and DNN-CRF-R, and the results are shown in Figure 1 and 2.

We document HIT−FA rates on three levels: overall, voiced intervals (pitched frames) and unvoiced intervals (unpitched frames). Voicing boundaries are determined using ideal pitch. Figure 1 shows the results for both matched-noise and unmatched-noise test conditions. First, comparing the performances of DNN-CRFs and DNNs, we can see that modeling temporal continuity always improves performance. It also seems very helpful for generalization to different SNRs. In the matched condition, the improvement by directly maximizing HIT−FA is most significant in unvoiced intervals. The improvement becomes larger when SNR decreases. In the unmatched condition, as classification becomes much harder, direct maximization of HIT−FA offers more improvements in all cases. The largest HIT−FA improvement of DNN-CRF*-R over DNN is about 10.7% and 21.2% absolute in overall and unvoiced speech intervals, respectively. For a closer inspection, Figure 2 shows channelwise HIT−FA comparisons on the 0 dB test mixtures in the matched-noise test condition. It is well known that unvoiced speech is indispensable for speech intelligibility but hard to separate. Due to the lack of harmonicity and weak energy, frequency channels containing unvoiced speech often have significantly skewed distributions of target-dominant and interference-dominant units. Therefore, an accuracy-maximizing classifier tends to output all 0's to attain a high accuracy. As an illustration, Figure 3 shows two masks for an utterance mixed with an unseen crowd noise at 0 dB using DNN and DNN-CRF*-P respectively. The two estimated masks achieve similar accuracy around 90%. However, it is clear that the DNN mask misses significant portions of unvoiced speech, e.g., between frame 30-50 and 220-240.

Table 1: Performance comparisons between different systems. Boldface indicates best result

| System | Matched-noise condition | | | | Unmatched-noise condition | | | |
|---|---|---|---|---|---|---|---|---|
| | Accuracy | HIT−FA | SNR (dB) | SegSNR (dB) | Accuracy | HIT−FA | SNR (dB) | SegSNR (dB) |
| GMM [10] | 77.4% | 55.4% | 10.2 | 7.3 | 65.9% | 31.6% | 6.8 | 1.9 |
| SVM [11] | 86.6% | 68.0% | 10.5 | 10.9 | **91.2%** | 64.1% | 9.7 | 7.9 |
| DNN | 87.7% | 71.6% | 11.4 | 11.8 | 91.1% | 66.2% | 9.9 | 8.1 |
| CRF | 82.3% | 59.8% | 8.8 | 8.7 | 90.8% | 64.0% | 9.3 | 7.8 |
| SVM-Struct | 81.7% | 58.6% | 8.4 | 8.1 | 90.7% | 63.5% | 9.1 | 7.5 |
| CNF | 87.8% | 71.7% | 11.2 | 12.0 | 91.1% | 66.9% | 9.8 | 8.4 |
| LD-CRF | 86.3% | 68.4% | 9.7 | 10.5 | 91.1% | 63.6% | 8.9 | 7.8 |
| DNN-CRF*-R | 89.1% | 75.6% | **12.1** | 13.2 | 90.8% | 70.2% | **10.3** | **9.0** |
| DNN-CRF*-P | **89.9%** | **76.9%** | 12.0 | **13.5** | 91.1% | **70.7%** | 10.0 | 8.9 |
| Hendriks et al. [1] | n/a | n/a | 4.6 | 0.5 | n/a | n/a | 6.2 | 1.1 |
| Wiener Filter [2] | n/a | n/a | 3.7 | -0.7 | n/a | n/a | 5.6 | -0.6 |

Table 2: Performance comparisons when tested on different unseen speakers

| System | Matched-noise condition | | | | Unmatched-noise condition | | | |
|---|---|---|---|---|---|---|---|---|
| | Accuracy | HIT−FA | SNR (dB) | SegSNR (dB) | Accuracy | HIT−FA | SNR (dB) | SegSNR (dB) |
| SVM [11] | 86.2% | 65.0% | 10.2 | 9.9 | **91.1%** | 60.6% | 9.4 | 7.3 |
| DNN-CRF*-P | **87.3%** | **72.0%** | **12.1** | **11.2** | 90.9% | **68.3%** | **10.1** | **8.1** |
| Hendriks et al. [1] | n/a | n/a | 4.5 | -2.9 | n/a | n/a | 6.9 | -1.0 |
| Wiener Filter [2] | n/a | n/a | 3.8 | -4.5 | n/a | n/a | 6.0 | -3.3 |

In summary, direct maximization of HIT−FA improves HIT−FA performance compared to accuracy maximization, especially for unvoiced speech, and the improvement is more significant when the system is tested on unseen acoustic environments.

## 4.3 Experiment 2: system comparisons

We systematically compare the proposed system with three kinds of systems on 0 dB mixtures: binary classifier based, structured predictor based, and speech enhancement based. In addition to HIT−FA, we also include classification accuracy, SNR and segmental SNR (segSNR) as alternative evaluation criteria. To compute SNRs, we use the target speech resynthesized from the IBM as the ground truth signal for all classification-based systems. This way of computing SNRs is commonly adopted in the literature [4, 25], as the IBM represents the ground truth of classification. All classification-based systems use the same feature set, but with estimated pitch, described in Section 2, except for Kim et al.'s GMM based system which uses AMS features [10]. Note that we fail to produce reasonable results using the complementary feature set in Kim et al.'s system, possibly because GMM requires much more training data than discriminative models for high dimensional features. Results are summarized in Table 1.

We first compare with methods based on binary classifiers. These include two existing systems [10, 11] and a DNN based system. Due to the variety of noises, classification is challenging even in the matched-noise condition. It is clear that the proposed system significantly outperforms the others in terms of all criteria. The improvement of DNN-CRF*s over DNN demonstrates the benefit of modeling temporal continuity. It is interesting to see that DNN significantly outperforms SVM, especially for unvoiced speech (not shown) which is important for speech intelligibility. We note that without RBM pretraining, DNN performs significantly worse. Classification in the unmatched-noise condition is obviously more difficult, as feature distributions are likely mismatched between the training and the test set. Kim et al.'s system fails to generalize to different acoustic environments due to substantially increased FA rates. The proposed system significantly outperforms SVM and DNN, achieving about 71% overall HIT−FA and 10 dB SNR for unseen noises. Kim et al.'s system has been shown to improve human speech intelligibility [10], it is therefore reasonable to project that the proposed system will provide further speech intelligibility improvements.

We next compare with systems based on structured predictors, including CRF, SVM-Struct [26], conditional neural fields (CNF) [20] and latent-dynamic CRF (LD-CRF) [19]. For fair comparisons, we use a two hidden layer CNF model with the same number of parameters as DNN-CRF*s. Conventional structured predictors such as CRF and SVM-Struct (linear kernel) are able to explicitly model temporal dynamics, but only with linear modeling capability. Direct use of CRF turns out to be much worse than using kernel SVM. Nevertheless, the performance can be substantially

boosted by adding latent variables (LD-CRF) or by using nonlinear feature functions (CNF and DNN-CRF*s). With the same network architecture, CNF mainly differs from our model in two aspects. First, CNF does not use unsupervised RBM pretraining. Second, CNF only uses bias units in building transition features. As a result, the proposed system significantly outperforms CNF, even if CRF and neural networks are jointly trained in the CNF model. With better ability of encoding contextual information, using a window of posteriors as features clearly outperforms single unit features in terms of classification. It is worth noting that although SVM achieves slightly higher accuracy in the unmatched-noise condition, the resulting HIT−FA and SNRs are worse than some other systems. This is consistent with our analysis in Section 4.2.

Finally, we compare with two representative speech enhancement systems [1, 2]. The algorithm proposed in [1] represents a recent state-of-the-art method and Wiener filtering [2] is one of the most widely used speech enhancement algorithms. Since speech enhancement does not aim to estimate the IBM, we compare SNRs by using clean speech (not the IBM) as the ground truth. As shown in Table 1, the speech enhancement algorithms are much worse, and this is true of all 17 noises.

Due to temporal continuity modeling and the use of T-F context, the proposed system produces masks that are smoother than those from the other systems (e.g., Figure 3). As a result, the outputs seem to contain less musical noise.

### 4.4 Experiment 3: speaker generalization

Although the training set contains only a single IEEE speaker, the proposed system generalizes reasonably well to different unseen speakers. To show this, we create a new test set by mixing 20 utterances from the TIMIT corpus [27] at 0 dB. The new test utterances are chosen from 10 different female TIMIT speakers, each providing 2 utterances. We show the results in Table 2, and it is clear that the proposed system generalizes better than existing ones to unseen speakers. Note that significantly better performance and generalization to different genders can be obtained by including the speaker(s) of interest into the training set.

## 5 Discussion and conclusion

Listening tests have shown that a high FA rate is more detrimental to speech intelligibility than a high miss (or low HIT) [9]. The proposed classification framework affords us control over these two quantities. For example, we could constrain the upper bound of the FA rate while still maximizing the HIT rate. In this case, a constrained optimization should substitute (5). Our experimental results (not shown due to lack of space) indicate that this can effectively remove spurious target segments while still produce intelligible speech.

Being able to efficiently compute the derivative of marginals, in principle one could optimize a class of objectives other than HIT−FA. These may include objectives concerning either speech intelligibility or quality, as long as the objective of interest can be expressed or approximated by a combination of marginal probabilities. For example, we have tried to simultaneously minimize two traditional CASA measures $P_{EL}$ and $P_{NR}$ (see e.g., [25]), where $P_{EL}$ represents the percent of target energy loss and $P_{NR}$ the percent of noise energy residue. Significant reductions in both measures can be achieved compared to methods that maximize accuracy or conditional log-likelihood.

We have demonstrated that the challenge of the monaural speech separation problem can be effectively approached via structured prediction. Observing that the IBM exhibits highly structured patterns, we have proposed to use CRF to explicitly model the temporal continuity in the IBM. This linear sequence classifier is further transformed to a nonlinear one by using state and transition feature functions learned from DNN. Consistent with the results from speech perception, we train the proposed DNN-CRF model to maximize a measure that is well correlated to human speech intelligibility in noise. Experimental results show that the proposed system significantly outperforms existing ones and generalizes better to different acoustic environments. Aside from temporal continuity, other ASA principles [5] such as common onset and co-modulation also contribute to the structure in the IBM, and we will investigate these in future work.

**Acknowledgements.** This research was supported in part by an AFOSR grant (FA9550-12-1-0130), an STTR subcontract from Kuzer, and the Ohio Supercomputer Center.

## Footnotes

[1]Test noises are: babble, bird chirp, crow, cocktail party, yelling, clap, rain, rock music, siren, telephone, white, wind, crowd, fan, speech shaped, traffic, and factory noise. The first 12 are used in training.

# References

[1] R. Hendriks, R. Heusdens, and J. Jensen, "MMSE based noise PSD tracking with low complexity," in *ICASSP*, 2010.

[2] P. Scalart and J. Filho, "Speech enhancement based on a priori signal to noise estimation," in *ICASSP*, 1996.

[3] S. Roweis, "One microphone source separation," in *NIPS*, 2001.

[4] D. Wang and G. Brown, Eds., *Computational Auditory Scene Analysis: Principles, Algorithms and Applications*. Hoboken, NJ: Wiley-IEEE Press, 2006.

[5] A.S. Bregman, *Auditory scene analysis: The perceptual organization of sound*. The MIT Press, 1994.

[6] D. Wang, "On ideal binary mask as the computational goal of auditory scene analysis," in *Speech Separation by Humans and Machines*, Divenyi P., Ed. Kluwer Academic, Norwell MA., 2005, pp. 181–197.

[7] D. Brungart, P. Chang, B. Simpson, and D. Wang, "Isolating the energetic component of speech-on-speech masking with ideal time-frequency segregation," *J. Acoust. Soc. Am.*, vol. 120, pp. 4007–4018, 2006.

[8] M. Anzalone, L. Calandruccio, K. Doherty, and L. Carney, "Determination of the potential benefit of time-frequency gain manipulation," *Ear and hearing*, vol. 27, no. 5, pp. 480–492, 2006.

[9] N. Li and P. Loizou, "Factors influencing intelligibility of ideal binary-masked speech: Implications for noise reduction," *J. Acoust. Soc. Am.*, vol. 123, no. 3, pp. 1673–1682, 2008.

[10] G. Kim, Y. Lu, Y. Hu, and P. Loizou, "An algorithm that improves speech intelligibility in noise for normal-hearing listeners," *J. Acoust. Soc. Am.*, vol. 126, pp. 1486–1494, 2009.

[11] K. Han and D. Wang, "An SVM based classification approach to speech separation," in *ICASSP*, 2011.

[12] Y. Wang, K. Han, and D. Wang, "Exploring monaural features for classification-based speech segregation," *IEEE Trans. Audio, Speech, Lang. Process.*, in press, 2012.

[13] G. Mysore and P. Smaragdis, "A non-negative approach to semi-supervised separation of speech from noise with the use of temporal dynamics," in *ICASSP*, 2011.

[14] J. Hershey, T. Kristjansson, S. Rennie, and P. Olsen, "Single channel speech separation using factorial dynamics," in *NIPS*, 2007.

[15] J. Lafferty, A. McCallum, and F. Pareira, "Conditional random fields: probabilistic models for segmenting and labeling sequence data," in *ICML*, 2001.

[16] J. Nocedal and S. Wright, *Numerical optimization*. Springer verlag, 1999.

[17] G. Hinton, S. Osindero, and Y. Teh, "A fast learning algorithm for deep belief nets," *Neural Computation*, vol. 18, no. 7, pp. 1527–1554, 2006.

[18] L. van der Maaten, M. Welling, and L. Saul, "Hidden-unit conditional random fields," in *AISTATS*, 2011.

[19] L. Morency, A. Quattoni, and T. Darrell, "Latent-dynamic discriminative models for continuous gesture recognition," in *CVPR*, 2007.

[20] J. Peng, L. Bo, and J. Xu, "Conditional neural fields," in *NIPS*, 2009.

[21] A. Mohamed, G. Dahl, and G. Hinton, "Deep belief networks for phone recognition," in *NIPS workshop on speech recognition and related applications*, 2009.

[22] T. Do and T. Artieres, "Neural conditional random fields," in *AISTATS*, 2010.

[23] L. Rabiner, "A tutorial on hidden Markov models and selected applications in speech recognition," *Proc. IEEE*, vol. 77, no. 2, pp. 257–286, 2003.

[24] IEEE, "IEEE recommended practice for speech quality measurements," *IEEE Trans. Audio Electroacoust.*, vol. 17, pp. 225–246, 1969.

[25] G. Hu and D. Wang, "Monaural speech segregation based on pitch tracking and amplitude modulation," *IEEE Trans. Neural Networks*, vol. 15, no. 5, pp. 1135–1150, 2004.

[26] I. Tsochataridis, T. Hofmann, and T. Joachims, "Support vector machine for interdependent and structured output spaces," in *ICML*, 2004.

[27] J. Garofolo, *DARPA TIMIT acoustic-phonetic continuous speech corpus*, NIST, 1993.

